# AverNet: All-in-one Video Restoration for Time-varying Unknown Degradations

**Haiyu Zhao**[1], **Lei Tian**[2], **Xinyan Xiao**[2], **Peng Hu**[1], **Yuanbiao Gou**[1,*], **Xi Peng**[1,3*]

[1]College of Computer Science, Sichuan University, China.
[2]Baidu Inc., Beijing, China.
[3]State Key Laboratory of Hydraulics and
Mountain River Engineering, Sichuan University, China.
{haiyuzhao.gm, penghu.ml, gouyuanbiao, pengx.gm}@gmail.com
{tianlei09, xiaoxinyan}@baidu.com

## Abstract

Traditional video restoration approaches were designed to recover clean videos from a specific type of degradation, making them ineffective in handling multiple unknown types of degradation. To address this issue, several studies have been conducted and have shown promising results. However, these studies overlook that the degradations in video usually change over time, dubbed time-varying unknown degradations (TUD). To tackle such a less-touched challenge, we propose an innovative method, termed as All-in-one VidEo Restoration Network (Aver-Net), which comprises two core modules, *i.e.*, Prompt-Guided Alignment (PGA) module and Prompt-Conditioned Enhancement (PCE) module. Specifically, PGA addresses the issue of pixel shifts caused by time-varying degradations by learning and utilizing prompts to align video frames at the pixel level. To handle multiple unknown degradations, PCE recasts it into a conditional restoration problem by implicitly establishing a conditional map between degradations and ground truths. Thanks to the collaboration between PGA and PCE modules, AverNet empirically demonstrates its effectiveness in recovering videos from TUD. Extensive experiments are carried out on two synthesized datasets featuring seven types of degradations with random corruption levels. The code is available at `https://github.com/XLearning-SCU/2024-NeurIPS-AverNet`.

## 1 Introduction

Video restoration aims to recover a high-quality video from a low-quality one that is corrupted by degradations such as noise, blur, and compression artifacts. Over the past few years, numerous studies [1, 2, 3, 4, 5] have been conducted, yielding promising performance in video restoration. However, these methods typically require prior knowledge of the specific type of degradation to design and train effective models, such as denoising, dehazing, and deblurring models [6, 7, 8, 9, 10, 11, 12]. In practice, obtaining such prior knowledge in advance is challenging, especially when the data is affected by multiple unknown types of degradation.

In the field of image restoration, efforts [13, 14, 15] have been made to restore images affected by multiple unknown types of degradation using a unified model, which is known as all-in-one image restoration (AIR) [16]. However, it is hard to achieve encouraging performance by simply applying existing AIR methods to videos due to the neglect of temporal information, which leads to inferior performance, as verified in our experiments. To favor all-in-one video restoration (AVR), a few

---

[*]Corresponding Authors

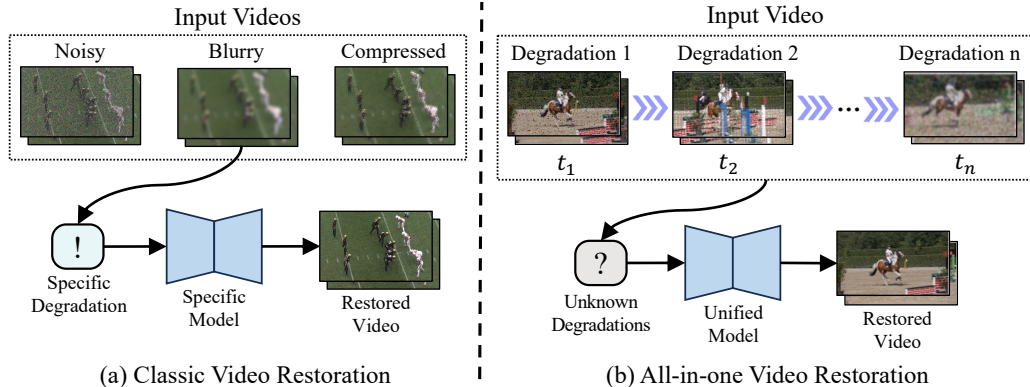

Figure 1: Illustration of classic and all-in-one video restoration. (a) aims to develop a specific model for each degradation to handle the corrupted video, assuming that the degradation types and levels are the same and known for all frames. In contrast, (b) intends to handle videos containing time-varying unknown degradations through a unified model, which is more practical and challenging.

studies [17, 18, 19] have been conducted in very recent, which typically assume that the frames of a given video contain unknown but the same type of degradation. In other words, these methods can only handle a single unknown type of degradation and would fail when faced with multiple unknown types of degradation for a given video. Clearly, the latter is more practical and challenging because degradations can vary over time in real-world scenarios (Fig. 1). For instance, motion blur can occur when a stationary object starts to move, and noise will appear when the scene changes to low-light conditions. Therefore, it is highly desirable to develop an AVR method capable of handling time-varying unknown degradations (TUD).

To overcome the aforementioned challenge of TUD, we propose an All-in-one VidEo Restoration Network (AverNet), which consists of two core modules, *i.e.*, Prompt-Guided Alignment (PGA) module and Prompt-Conditioned Enhancement (PCE) module. To recover videos from time-varying degradations, PGA is designed based on the following observation. Specifically, a vital step in video restoration is spatial alignment across frames, which aims to eliminate frame difference at the pixel level. As shown in Fig. 2, compared to time-invariant degradations, time-varying degradations cause larger and more complex pixel shifts, making frame alignment significantly more challenging. Hence, to align frames at pixel level better, PGA optimizes a specific prompt conditioned on a video clip first and then uses this prompt to align the frames within the clip. To tackle unknown degradations in a given frame, PCE is designed to learn prompts corresponding to latent unknown degradations by building the conditional map between the corrupted frame and the ground truth. Through treating the prompts as the conditions, PCE transforms the task of video restoration from multiple unknown degradations into a known conditional restoration problem.

To summarize, the contributions of this work are as below:

- As far as we know, this work could be the first study on the TUD. Different from previous efforts on time-invariant degradation, our method is suitable for more practical scenarios wherein the types and levels of degradations are both unknown and changing over time.

- We show that the TUD challenge could be effectively addressed by our AverNet. In brief, AverNet employs the PGA module to address the issue of the large and complex pixel shifts and the PCE module to solve the problem of multiple unknown degradations.

- Although the TUD setting is practical, the data with ground-truth is scarce and even impossible to collect. To address the data scarcity issue, we develop a data synthesis approach to simulate the data of TUD in real world. Experiments on two datasets across seven types of degradations with random levels demonstrate the effectiveness of our AverNet.

## 2 Related Work

In this section, we introduce the related works in video restoration and all-in-one image restoration, and elaborate on the differences between our AverNet and the existing methods.

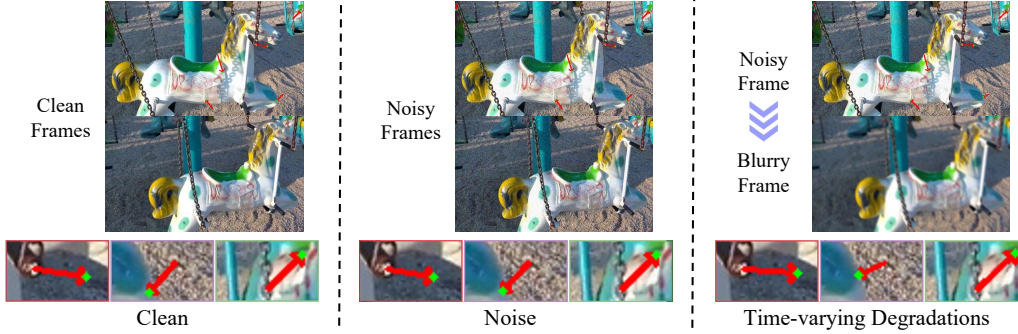

Figure 2: Illustration of the pixel shift issue. We compute the optical flow between two consecutive frames with time-invariant and time-varying degradations. Several directional vectors are visualized as red arrows to indicate the estimated pixel alignments between the two frames. One could observe that time-varying degradations lead to less accurate estimations compared to time-invariant degradations, causing a larger and more complex pixel shift after alignment.

## 2.1 Video Restoration

Currently, deep video restoration methods could be roughly divided into two categories according to their architectures, *i.e.*, sliding window-based methods [1, 5, 20, 21, 22, 23, 19] and recurrent methods [24, 25, 26, 2, 4]. The former kind of methods typically take a short sequence of frames as input and restore only the center frame. By leveraging the temporal information from adjacent frames, these methods have achieved promising performance. For example, EDVR [1] introduced the pyramid architecture and temporal-spatial attention modules to effectively aggregate information across frames. Shift-Net [5] proposed an efficient framework based on grouped spatial-temporal shift modules to implicitly aggregate inter-frame information. Although the methods have shown impressive performance, they encounter challenges in handling long-sequence videos and suffer from high memory consumption. To alleviate the problem, recurrent methods choose to propagate latent features from one frame to the next frame sequentially, accumulating information from previous frames for the restoration of subsequent frames. For instance, BasicVSR [26] proposed a concise and efficient network employing a bidirectional propagation scheme. BasicVSR++ [2] improved the bidirectional scheme with second-order grid connections to implement an effective recurrent model.

Different from the above methods focusing on a specific degradation, ViWS-Net [17], CUDN [18] and Diff-TTA [19] explored a more generalized task of AVR, which aims to handle multiple unknown degradations through a unified model. Specifically, ViWS-Net introduces degradation messenger tokens to learn specific degradation information and employs them to guide the restoration. CUDN adaptively estimates the features of unknown degradations and employs the estimation to guide the model to remove diverse degradations. Diff-TTA introduces test-time adaptation techniques [27] to adapt the distribution of test data and recalibrates the parameters of pre-trained models to address unknown degradations. Although there is a similarity between AverNet and these methods in the task of AVR, they are remarkably different in both problem and solution. In problem, ViWS-Net, CUDN and Diff-TTA considered the unknown types of degradations, while AverNet considers not only the unknown types and levels of degradations, but also their variations over time, which is more practical and challenging. In solution, they follow the paradigm of sliding window-based methods and employ local information for restoration. In contrast, AverNet improves the propagation scheme of the recurrent paradigm to address TUD challenge, enabling it to effectively utilize global temporal information for AVR.

## 2.2 All-in-one Image Restoration

Some recent works [13, 28, 29, 30, 31] have been devoted to all-in-one image restoration (AIR), which aims to recover images from multiple unknown types of degradations using a single model. The key to these methods is extracting discriminative information for different degradations, and employing this information to guide a single model for performing AIR. For example, AirNet [16] learns the degradation representations through contrastive learning, and uses the representations to guide the restoration network. PromptIR [13] proposes to use learnable prompts to encode

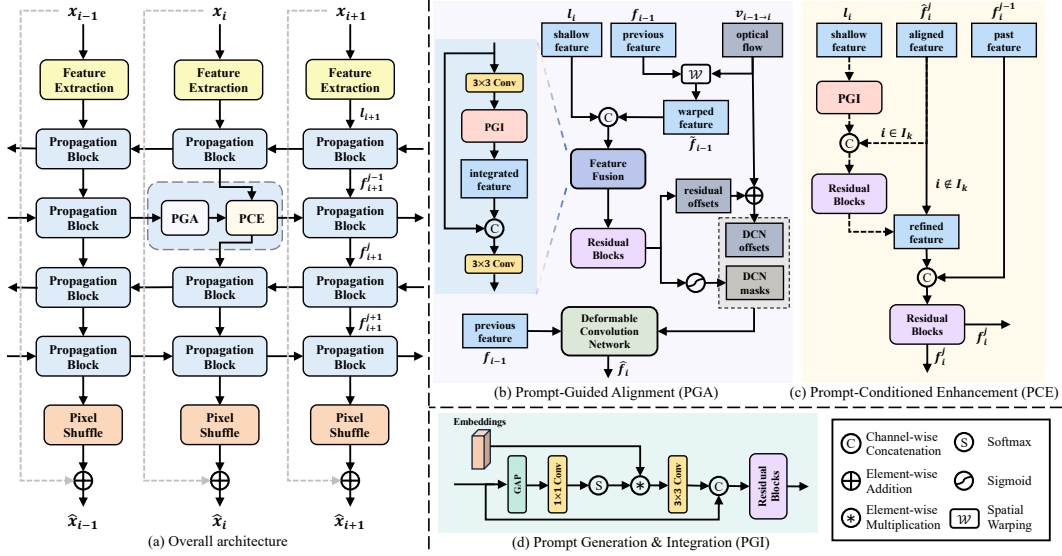

Figure 3: Architecture overview. (a) Overall architecture of our AverNet, which is mainly composed of propagation blocks. Each block consists of a (b) PGA module for spatially aligning features across frames with time-varying degradations, and a (c) PCE module for enhancing the features of current frame with unknown degradations. (d) PGI modules endow PGA and PCE with the capacity of conditioning on degradations by means of input-conditioned prompts. For simplicity, the superscripts $j$ in (b) are omitted. In (c), the past feature is from the last time of propagation, and $I_k$ refers to the indices of key frames.

degradation-specific information first and then guide the restoration of clean images. ProRes [32] introduces additional visual prompts to incorporate task-specific information and utilize the prompts to guide the network for AIR. MPerceiver [33] proposes a multimodal prompt learning approach to exploit the generative priors of Stable Diffusion [34] to achieve high-fidelity all-in-one image restoration. AutoDIR [14] introduces a CLIP-based module to detect unknown degradations, and instructs a latent-diffusion-based module with a text prompt for restoration. In addition to AIR, a more recent work TAO [35] has explored a challenging task of open-set image restoration, which aims to handle unknown degradations that were unforeseen in the pretraining phase.

Unlike the above methods specialized for image restoration, our AverNet is devoted to all-in-one video restoration, which aims to recover videos affected by time-varying unknown degradations using a single model. In contrast to AIR focusing on unknown degradations, our AVR is more practical and challenging since the degradations are both unknown and time-varying.

## 3    Proposed Method

In this section, we elaborate on our AverNet and two modules, *i.e.*, Prompt-Guided Alignment module (PGA) and Prompt-Conditioned Enhancement module (PCE).

The architecture of our AverNet is depicted in Fig. 3. Given an input video, residual blocks are first applied to extract shallow features from each frame. Then, the features are sequentially propagated four times across the video sequence to incorporate temporal information. Specifically, the first and third times of propagation are backward, in which the features are propagated from the last frame to the first one. The second and fourth times of propagation are forward, in which the features are propagated from the first frame to the last one. Such an alternating bidirectional propagation enables the features of each frame to involve abundant information from the whole video sequence for restoration. After propagation, the features are used to generate the final output for each frame through convolutions and pixel-shuffling [36] blocks. In the following, we will detail our core designs, *i.e.*, PGA and PCE, for solving TUD challenge during the propagation process.

### 3.1 Prompt-Guided Alignment

To address the pixel shift issue and obtain aligned features from the video with time-varying degradations, PGA employs deformable convolution network [37] (DCN) to align the spatial contents between frames. The calculation of DCN depends on two parameters of offsets and masks, which indicate spatial coordinates shifts and weights, respectively. Typically, the offsets and masks are estimated from the frame features and optical flow [38] through convolutions. However, the time-varying degradations lead to inaccurate estimations and deteriorate the spatial alignments. Therefore, we further introduce the input-conditioned prompts that encode degradation information to guide the estimation. The graphical illustration is shown in Fig. 3(b). In the rest of this section, we will detail the alignment procedure in the forward propagation, which is similar in the backward propagation. For simplicity, we omit the superscript $j$ which indicates the $j$-th time of propagation.

Given the shallow features $l_i$ of the $i$-th frame, the previous features $f_{i-1}$ from $(i-1)$-th frame, and the optical flow $v_{(i-1)\to i}$ from the $(i-1)$-th to $i$-th frame, we first warp $f_{i-1}$ through $v_{(i-1)\to i}$:

$$\tilde{f}_{i-1} = \mathcal{W}(f_{i-1}, v_{(i-1)\to i}), \tag{1}$$

where $\mathcal{W}$ denotes the spatial warping function. Then, we concatenate the shallow features $l_i$ and the warped features $\tilde{f}_{i-1}$ into a clip $\tilde{f}_i$ to estimate the offsets and masks of DCN. Instead of directly estimating them through a simple network, we introduce prompts to encode degradation-specific information conditioned on the clip, and integrate prompts into the clip to guide the alignment of the frames. Specifically, the integrated features $g_i$ are generated by a convolution followed by a Prompt Generation & Integration (PGI) block (Fig. 3(d)):

$$g_i = PGI(\text{Conv}_{3\times3}(\tilde{f}_i)), \tag{2}$$

where $\tilde{f}_i = Concat(l_i, \tilde{f}_{i-1})$ and $Concat(\cdot)$ refers to channel concatenation. Specifically, PGI obtains the integrated features by generating input-conditioned prompts and fusing them with the input features. Taking $x$ as the input features, PGI first predicts attention-based weights $W \in \mathbb{R}^N$ and then applies them to prompt embeddings $E \in \mathbb{R}^{N\times\hat{C}\times\hat{H}\times\hat{W}}$ to obtain input-conditioned prompts $P$. To be specific, PGI sequentially applies global average pooling (GAP), $1 \times 1$ convolution, and softmax operation on $x$ to obtain the weights $W$:

$$W = \text{Softmax}(\text{Conv}_{1\times1}(\text{GAP}(x))). \tag{3}$$

After that, $P$ is calculated as the weighted sum of $E$ followed by $3 \times 3$ convolution for refinement:

$$P = \text{Conv}_{3\times3}(\sum_{n=1}^{N} W_n \cdot E_n). \tag{4}$$

Note that $P$ has a fixed spatial size $\hat{H} \times \hat{W}$, which may be different from the size of frame features. Therefore, we apply the bilinear upsampling operation to upscale $P$ to the same size as the input features. Finally, the prompts are integrated into the input features $x$ through:

$$g_i = \mathcal{R}(Concat(x, P)), \tag{5}$$

where $\mathcal{R}(\cdot)$ denotes residual blocks. After obtaining the integrated features $g_i$, we combine them with the input $\tilde{f}_i$ and refine them using a $3 \times 3$ convolution:

$$h_i = \text{Conv}_{3\times3}(Concat(\tilde{f}_i, g_i)), \tag{6}$$

where $h_i$ is the intermediate features to compute DCN parameters. Finally, the DCN offsets $o_{(i-1)\to i}$ and modulation masks $m_{(i-1)\to i}$ are calculated as:

$$\begin{aligned} o_{(i-1)\to i} &= v_{(i-1)\to i} + \mathcal{R}^o(h_i), \\ m_{(i-1)\to i} &= \sigma(\mathcal{R}^m(h_i)), \end{aligned} \tag{7}$$

where $R^{\{o,m\}}$ denotes a stack of convolutions, and $\sigma$ denotes the sigmoid function. A DCN is then applied to the previous feature $f_{i-1}$ for spatial alignment:

$$\hat{f}_i = \mathcal{D}(f_{i-1}; o_{(i-1)\to i}, m_{(i-1)\to i}), \tag{8}$$

where $\mathcal{D}(\cdot)$ denotes a deformable convolution network.

## 3.2 Prompt-Conditioned Enhancement

Here, we detail PCE module in the forward propagation, which is similar in the backward propagation. To handle multiple unknown degradations, PCE employs the prompts corresponding to the latent unknown degradations as the conditions to enhance the current frame features during propagation. However, the prompt extraction and conditional enhancement require additional computations which are non-trivial. To reduce the computational cost, PCE only conducts enhancement at selected key frames and collaterally enhances the features of other frames through propagation. Specifically, key frames are sparsely selected based on a fixed interval $T$. The indices of key frames can be expressed as a list $I_k$, which is an arithmetic sequence with a common difference of $T$. During the propagation of key frames, PCE enhances the features based on the aligned features $\hat{f}_i^j$ and the shallow features $l_i$ of key frames.

To obtain the enhanced features $f_i^j$, PCE first employs PGI to extract the key frame prompt $k_i$ based on the shallow features $l_i$ according to the procedures shown in Fig. 3(d). Then, the key frame prompt is used as the condition to enhance aligned features $\hat{f}_i^j$ through a few residual blocks:

$$\overline{f}_i^j = \mathcal{R}(Concat(\hat{f}_i^j, k_i)), \quad i \in I_k, \tag{9}$$

where $\mathcal{R}(\cdot)$ denotes residual blocks, $Concat(\cdot)$ denotes concatenation along channel dimension, and $I_k$ refers to the list of indices of key frames. After that, the enhanced features are fused with the propagation features of last time for the subsequent propagation:

$$f_i^j = \overline{f}_i^j + \mathcal{R}(Concat(f_i^{j-1}, \overline{f}_i^j)), \quad i \in I_k, \tag{10}$$

where $f_i^j$ and $f_i^{j-1}$ are the propagation features of current and last time at the $i$-th frame, respectively.

In the cases without enhancement, *i.e.*, the frame index $i \notin I_k$, the aligned features are directly concatenated with the propagation features of last time and then refined through residual blocks to obtain propagation features $f_i^j$:

$$f_i^j = \hat{f}_i^j + \mathcal{R}(Concat(f_i^{j-1}, \hat{f}_i^j)), \quad i \notin I_k. \tag{11}$$

Note that $f_i^0 = l_i$ for the first time of propagation.

## 4 Experiment

In this section, we conduct experiments to evaluate our AverNet. In the following, we introduce the experimental settings first and then show quantitative and qualitative results. Finally, we conduct ablation analyses to demonstrate the effectiveness of our designs.

### 4.1 Experiment Settings

**Video Synthesis Approach.** Although the TUD setting is practical, the data with ground-truth is scarce and even impossible to collect. To tackle this issue, we develop a data synthesis approach to simulate such pairs based on the degradation models in [39, 40]. To be specific, a clean video is first cut into multiple clips with a fixed interval $t$. Then, for each clip, a series of degradations are sampled from seven candidate degradations with a probability of $0.55$. Specifically, the candidate degradations include Poisson noise, Gaussian noise, speckle noise, resizing blur, Gaussian blur, JPEG compression, and video compression. The details for each degradation are provided in the appendix. After that, the degradations are shuffled and added to the clip to produce a corrupted one. Finally, the corrupted clips are assembled into a corrupted video that contains time-varying unknown degradations.

**Video Datasets.** We adopt two widely-used video datasets in experiments, *i.e.*, DAVIS [41] and Set8 [42]. Specifically, DAVIS contains 90 training sequences and 30 test sequences of resolution $854 \times 480$. Set8 has 8 test video sequences of resolution $960 \times 540$. We train all models on DAVIS training set, and test them on DAVIS-test and Set8. The training and test pairs are constructed through the above video synthesis approach. For the training set, the interval $t$ of degradation variations is set to 6 and the degradations are the above seven degradations including three types of noise, two types of blur, and two types of compression. For the test set, we apply different settings to generate diverse sets for full evaluations, which are detailed in Sec. 4.2.

**Implementation Details.** We use the same settings for all experiments. To be specific, the number of channels is set to 96, and the embedding length, dimension, and size of prompts are set to 5, 96, and 96×96, respectively. For optical flow estimation, we use the pre-trained SPyNet [43, 44] whose parameters and runtime are included in our AverNet.

**Training Details.** The experiments are conducted in PyTorch [45] framework with four NVIDIA GeForce RTX 3090 GPUs. For training, we use Charbonnier loss [46] and Adam [47] optimizer with $\beta_1 = 0.9$ and $\beta_2 = 0.999$. The initial learning rates of main and optical flow networks are set to $1e^{-4}$ and $2.5e^{-5}$, respectively, which are gradually decreased to $1e^{-7}$ through cosine annealing strategy [48]. The number and resolution of input frames are set to 12 and $256 \times 256$, respectively. We train the networks with the batch size of 1 for 600K iterations, in which the parameters of optical flow network will not be updated for the first 5K iterations.

## 4.2 Comparison Experiments

In this section, we compare our AverNet with existing state-of-the-art methods on the video datasets with TUD. To be specific, existing methods include four representative all-in-one image restoration methods and four conventional video restoration methods. The all-in-one image restoration methods are WDiffusion [49], TransWeather [50], PromptIR [13], and AirNet [16]. The video restoration methods are EDVR [1], BasicVSR++ [2], Shift-Net [5], and RVRT [4]. Both the video and image restoration methods were trained on DAVIS training set from scratch according to the training settings in their papers. Note that the image restoration methods were trained and tested on each frame of the video sequences. We present the parameters and runtime of our AverNet and the compared methods in Tab. 1, which are computed according to their original test settings. Specifically, the runtime was computed on a video with 48 frames from DAVIS-test.

Table 1: Comparisons on parameters and runtime. From the table, one could observe that our method is more efficient than most AIR methods.

| Method | Video Restoration | | | | All-in-one Image Restoration | | | | AVR |
|---|---|---|---|---|---|---|---|---|---|
| | EDVR | BasicVSR++ | Shift-Net | RVRT | WDiffusion | TransWeather | AirNet | PromptIR | Ours |
| #Param | 23.6M | 7.4M | 12.9M | 13.6M | 80.0M | 38.1M | 8.9M | 35.6M | 41.3M |
| Runtime | 6.78s | 5.09s | 45.54s | 15.07s | 3956.50s | 3.32s | 12.98s | 10.81s | 6.93s |

To comprehensively evaluate AverNet on dealing with TUD, we conduct experiments on time-varying unknown degradations with different variation intervals and different degradation combinations. To be specific, the variation interval refers to the interval of video degradation variations over time and degradation combinations refer to various types of degradations in the video.

**Evaluation on Different Variation Intervals.** To evaluate the effectiveness in handling TUD, we test all models on six test sets that are generated through the video synthesis approach, where the variation interval is $t$. Specifically, six test sets are synthesized with the variation intervals $t = 6, 12, 24$ based on DAVIS-test and Set8, respectively.

As shown in Tab. 2, our method nearly outperforms other methods across all six test sets. For example, our method outperforms RVRT by 0.08dB~0.18dB in PSNR on DAVIS-test and outperforms Shift-Net by up to 0.99dB and 0.65dB on DAVIS-test and Set8, respectively. Additionally, as shown in Tab. 1, our method achieves the best performance while maintaining the highest efficiency. Specifically, compared with our method, the runtime of RVRT is more than double, and that of Shift-Net is over six times longer. In comparison with the all-in-one image restoration methods, our method yields significantly better results, showing the importance of leveraging temporal information in videos. For example, our AverNet outperforms AirNet by at least 1.53dB/0.0428 and 1.02dB/0.0345 in PSNR/SSIM on DAVIS-test and Set8, respectively, while requiring nearly half the runtime.

Fig. 4 and Fig. 5 present the qualitative results on DAVIS and Set8 datasets with interval $t = 12$, respectively. From the figures, one could observe that all-in-one image restoration methods like PromptIR and AirNet yield distorted and blurry results. Furthermore, BasicVSR++ and RVRT exhibit residual noise and artifacts. In contrast, our method excels in recovering structures and capturing finer details, resulting in clearer restorations.

Table 2: Quantitative results compared to state-of-the-art methods on test sets with various variation intervals. $t$ is the interval of degradation variations in the frame sequences. The best outcomes are highlighted in **bold**. From the table, one could observe that our method almost outperforms other methods on all test sets.

| Method | DAVIS-test | | | | | | Set8 | | | | | |
|---|---|---|---|---|---|---|---|---|---|---|---|---|
| | t=6 | | t=12 | | t=24 | | t=6 | | t=12 | | t=24 | |
| | PSNR | SSIM | PSNR | SSIM | PSNR | SSIM | PSNR | SSIM | PSNR | SSIM | PSNR | SSIM |
| WDiffusion | 31.74 | 0.8768 | 31.79 | 0.8784 | 31.92 | 0.8809 | 30.31 | 0.8784 | 30.02 | 0.8716 | 30.82 | 0.8746 |
| TransWeather | 31.11 | 0.8694 | 31.13 | 0.8699 | 31.26 | 0.8741 | 29.24 | 0.8662 | 28.95 | 0.8565 | 29.15 | 0.8632 |
| AirNet | 32.46 | 0.8873 | 32.46 | 0.8887 | 32.75 | 0.8928 | 30.71 | 0.8874 | 30.40 | 0.8806 | 31.16 | 0.8825 |
| PromptIR | 32.18 | 0.8843 | 32.19 | 0.8867 | 32.45 | 0.8900 | 30.79 | 0.8903 | 30.43 | 0.8821 | 31.19 | 0.8847 |
| EDVR | 28.70 | 0.7224 | 28.37 | 0.6991 | 29.07 | 0.7289 | 26.75 | 0.7259 | 26.94 | 0.7382 | 28.71 | 0.7675 |
| BasicVSR++ | 33.22 | 0.9204 | 33.07 | 0.9180 | 33.32 | 0.9210 | 30.90 | 0.9048 | 30.52 | 0.8965 | 31.35 | 0.9011 |
| Shift-Net | 33.09 | 0.9096 | 33.10 | 0.9113 | 33.34 | 0.9133 | 31.15 | 0.9027 | 30.82 | 0.8947 | 31.88 | 0.9000 |
| RVRT | 33.99 | 0.9314 | 33.98 | 0.9311 | 34.10 | 0.9315 | 31.73 | 0.9192 | 31.39 | 0.9113 | **32.47** | 0.9178 |
| AverNet (Ours) | **34.07** | **0.9333** | **34.09** | **0.9339** | **34.28** | **0.9356** | **31.73** | **0.9219** | **31.47** | **0.9145** | 32.45 | **0.9189** |

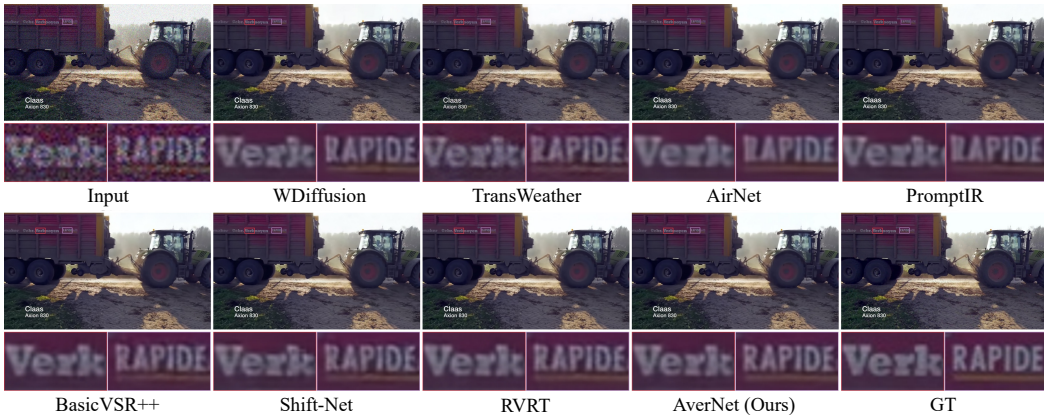

Figure 4: Qualitative results on the "tractor" video from DAVIS-test ($t = 12$), from which one could observe that existing methods leave residual noise or artifacts in the results. In contrast, our method obtains clearer results that are closer to GT.

**Evaluation on Different Degradation Combinations.** To evaluate the robustness in the situation with different degradation combinations, we construct new test sets by removing one type of degradation from the seven candidate degradations in the video synthesis. For example, we construct a test set with a new combination of noise and blur degradations by removing JPEG compression and video compression in the seven degradations. As a result, we obtain three different combinations, *i.e.*, noise&blur, noise&compression, and blur&compression.

From Tab. 1 and 3, one could observe that our method obtains comparable or even better performance while embracing much higher efficiency. For example, our method outperforms RVRT by 0.34dB and 0.24dB in PSNR on the DAVIS-test and Set8, respectively, with the noise&blur combination. Note that our method only requires less than half the time of RVRT to process the videos. Besides, our method outperforms Shift-Net by at most 1.01dB and 0.47dB in PSNR on DAVIS-test and Set8, respectively. Compared with all-in-one image restoration methods, our method has superior performance in every degradation combination on the test sets.

## 4.3 Ablation Experiments

To investigate the effectiveness of our AverNet, we conduct ablation experiments on the core modules, namely, PGA and PCE. All experiments were conducted on the test sets with $t = 12$.

To verify the effectiveness of PGA and PCE, we replace each of them with conventional propagation and alignment modules, respectively, while excluding the guidance and the condition from prompts. The results are presented in Tab. 4. In the table, it is apparent that each component brings considerable improvement, with PSNR gains ranging from 0.16dB to 2.15dB on the two test sets. Specifically,

Table 3: Quantitative results compared to state-of-the-art methods in three degradation combinations, *i.e.*, noise&blur, noise&compression, and blur&compression. The best outcomes are highlighted in **bold**. From the table, one could see that our method outperforms other methods in SSIM, and obtains comparable PSNR values to RVRT while requiring only half the runtime.

| Method | DAVIS-test | | | | | | Set8 | | | | | |
|---|---|---|---|---|---|---|---|---|---|---|---|---|
| | Noise & Blur | | Noise & Comp. | | Blur & Comp. | | Noise & Blur | | Noise & Comp. | | Blur & Comp. | |
| | PSNR | SSIM | PSNR | SSIM | PSNR | SSIM | PSNR | SSIM | PSNR | SSIM | PSNR | SSIM |
| WDiffusion | 32.70 | 0.8990 | 33.52 | 0.9124 | 33.76 | 0.9142 | 31.64 | 0.8943 | 30.88 | 0.8968 | 31.22 | 0.8978 |
| TransWeather | 31.74 | 0.8863 | 32.53 | 0.9062 | 32.18 | 0.9017 | 29.74 | 0.8714 | 29.93 | 0.8886 | 29.61 | 0.8830 |
| AirNet | 33.41 | 0.9078 | 34.23 | 0.9184 | 34.59 | 0.9224 | 32.15 | 0.9065 | 31.27 | 0.9019 | 31.60 | 0.9027 |
| PromptIR | 33.69 | 0.9128 | 34.18 | 0.9213 | 33.87 | 0.9179 | 32.10 | 0.9033 | 31.54 | 0.9106 | 31.53 | 0.9047 |
| EDVR | 28.00 | 0.6809 | 29.58 | 0.7036 | 34.17 | 0.9082 | 27.82 | 0.7268 | 27.23 | 0.7245 | 32.15 | 0.8845 |
| BasicVSR++ | 33.89 | 0.9324 | 34.72 | 0.9391 | 34.82 | 0.9392 | 31.88 | 0.9189 | 31.42 | 0.9152 | 31.35 | 0.9146 |
| Shift-Net | 34.00 | 0.9277 | 34.91 | 0.9390 | 35.26 | 0.9376 | 32.66 | 0.9159 | 31.86 | 0.9184 | 32.08 | 0.9164 |
| RVRT | 34.67 | 0.9438 | 35.69 | 0.9504 | **35.94** | 0.9503 | 32.70 | 0.9291 | **32.40** | 0.9297 | **32.38** | 0.9291 |
| AverNet (Ours) | **35.01** | **0.9468** | **35.89** | **0.9531** | 35.87 | **0.9506** | **32.94** | **0.9326** | 32.33 | **0.9309** | 32.37 | **0.9306** |

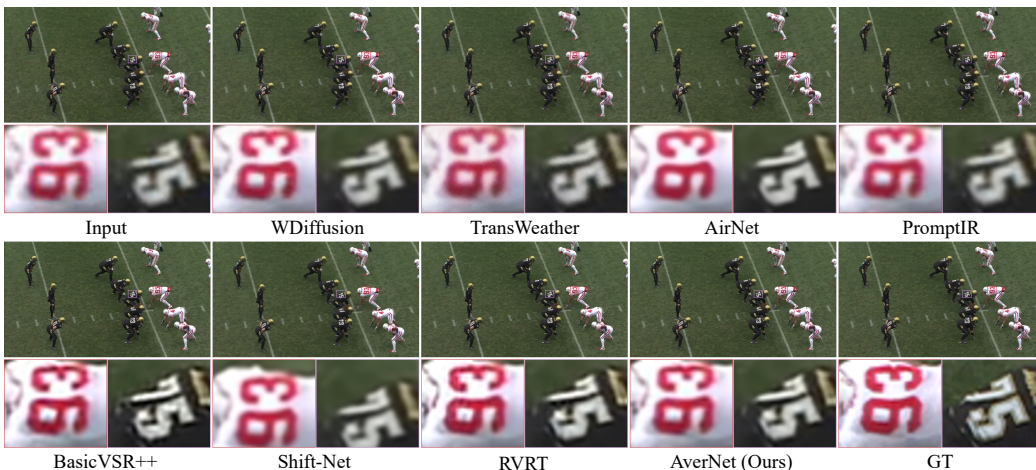

Figure 5: Qualitative results on the "touchdown" video from Set8 ($t = 12$), from which one could see that existing methods yield blurry or distorted results. In contrast, the results of our method have sharper outlines and less artifacts.

the model without PCE suffers from PSNR drops of 1.10dB and 1.67dB on DAVIS-test and Set8, respectively. Similarly, the model without PGA exhibits PSNR drops of 1.50dB and 1.33dB on the two test sets. Notably, as Set8 contains longer videos, the propagation errors caused by multiple unknown degradations are more serious. By enhancing the propagated features of key frames, PCE effectively mitigates these errors and makes significant contributions to the final performance. Additionally, the model without both PGA and PGE shows a significant PSNR drops of 1.66dB and 3.48dB, highlighting the effectiveness of prompt guidance and prompt conditioning.

Table 4: Ablation studies of the modules. Each module brings improvements in PSNR and SSIM, verifying their effectiveness.

| | PGA | PCE | DAVIS-test | | Set8 | |
|---|---|---|---|---|---|---|
| | | | PSNR | SSIM | PSNR | SSIM |
| (A) | | | 32.43 | 0.8910 | 27.99 | 0.8404 |
| (B) | | ✓ | 32.59 | 0.9157 | 30.14 | 0.8958 |
| (C) | ✓ | | 32.99 | 0.9156 | 29.80 | 0.8755 |
| (D) | ✓ | ✓ | 34.09 | 0.9339 | 31.47 | 0.9145 |

To investigate the influences of intervals between key frames, we change $T$ from 6, 12, to 24 and present the results in Tab. 5. One could observe that PCE with varied $T$ obtains similar results on

DAVIS-test while PCE with larger $T$ suffers from 0.45dB drop in PSNR on Set8. This is because the Set8 contains much longer videos compared to DAVIS-test. Consequently, the accumulated propagation errors on Set8 are more serious. This result further shows the importance of our PCE module. According to the results, we find $T = 6$ is a suitable value and set the interval of key frames to 6 in all experiments.

Table 5: Ablation of the key frame interval $T$ in PCE. From the table, one could see that larger $T$, *i.e.*, fewer key frames result in worse performance on the long videos of Set8.

| Datasets | DAVIS-test | | | Set8 | | |
|---|---|---|---|---|---|---|
| Intervals | T=6 | T=12 | T=24 | T=6 | T=12 | T=24 |
| PSNR | 34.09 | 34.15 | 34.14 | 31.47 | 31.02 | 31.02 |
| SSIM | 0.9339 | 0.9349 | 0.9349 | 0.9145 | 0.9066 | 0.9066 |

## 5 Conclusions

In this paper, we study a practical and challenging problem in video restoration, *i.e.*, time-varying unknown degradations. To solve the problem, we propose AverNet which could recover clean video from the corrupted ones with TUD. Different from existing video restoration methods, AverNet assumes the degradations are time-varying and could handle TUD without the prior of degradation. Extensive experimental results show the superiority of AverNet in both quantitative and qualitative comparisons.

## 6 Acknowledgment

This work was supported in part by the Fundamental Research Funds for the Central Universities under Grant CJ202303; in part by NSFC under Grant U21B2040; in part by Sichuan Science and Technology Planning Project under Grant 2024NSFTD0038.

## References

[1] Xintao Wang, Kelvin CK Chan, Ke Yu, Chao Dong, and Chen Change Loy. Edvr: Video restoration with enhanced deformable convolutional networks. In *Proceedings of the IEEE/CVF conference on computer vision and pattern recognition workshops*, pages 0–0, 2019.

[2] Kelvin CK Chan, Shangchen Zhou, Xiangyu Xu, and Chen Change Loy. Basicvsr++: Improving video super-resolution with enhanced propagation and alignment. In *Proceedings of the IEEE/CVF conference on computer vision and pattern recognition*, pages 5972–5981, 2022.

[3] Jingyun Liang, Jiezhang Cao, Yuchen Fan, Kai Zhang, Rakesh Ranjan, Yawei Li, Radu Timofte, and Luc Van Gool. Vrt: A video restoration transformer. *arXiv preprint arXiv:2201.12288*, 2022.

[4] Jingyun Liang, Yuchen Fan, Xiaoyu Xiang, Rakesh Ranjan, Eddy Ilg, Simon Green, Jiezhang Cao, Kai Zhang, Radu Timofte, and Luc V Gool. Recurrent video restoration transformer with guided deformable attention. *Advances in Neural Information Processing Systems*, 35:378–393, 2022.

[5] Dasong Li, Xiaoyu Shi, Yi Zhang, Ka Chun Cheung, Simon See, Xiaogang Wang, Hongwei Qin, and Hongsheng Li. A simple baseline for video restoration with grouped spatial-temporal shift. In *Proceedings of the IEEE/CVF Conference on Computer Vision and Pattern Recognition*, pages 9822–9832, 2023.

[6] Jing Lin, Yuanhao Cai, Xiaowan Hu, Haoqian Wang, Youliang Yan, Xueyi Zou, Henghui Ding, Yulun Zhang, Radu Timofte, and Luc Van Gool. Flow-guided sparse transformer for video deblurring. In *International Conference on Machine Learning*, pages 13334–13343. PMLR, 2022.

[7] Jing Lin, Xiaowan Hu, Yuanhao Cai, Haoqian Wang, Youliang Yan, Xueyi Zou, Yulun Zhang, and Luc Van Gool. Unsupervised flow-aligned sequence-to-sequence learning for video restoration. In *International Conference on Machine Learning*, pages 13394–13404. PMLR, 2022.

[8] Zichun Wang, Yulun Zhang, Debing Zhang, and Ying Fu. Recurrent self-supervised video denoising with denser receptive field. In *Proceedings of the 31st ACM International Conference on Multimedia*, pages 7363–7372, 2023.

[9] Jiezhang Cao, Qin Wang, Jingyun Liang, Yulun Zhang, Kai Zhang, and Luc Van Gool. Practical real video denoising with realistic degradation model. 2022.

[10] Jinshan Pan, Haoran Bai, and Jinhui Tang. Cascaded deep video deblurring using temporal sharpness prior. In *Proceedings of the IEEE/CVF conference on computer vision and pattern recognition*, pages 3043–3051, 2020.

[11] Shangchen Zhou, Jiawei Zhang, Jinshan Pan, Haozhe Xie, Wangmeng Zuo, and Jimmy Ren. Spatio-temporal filter adaptive network for video deblurring. In *Proceedings of the IEEE/CVF international conference on computer vision*, pages 2482–2491, 2019.

[12] Senyou Deng, Wenqi Ren, Yanyang Yan, Tao Wang, Fenglong Song, and Xiaochun Cao. Multi-scale separable network for ultra-high-definition video deblurring. In *Proceedings of the IEEE/CVF International Conference on Computer Vision*, pages 14030–14039, 2021.

[13] Vaishnav Potlapalli, Syed Waqas Zamir, Salman Khan, and Fahad Shahbaz Khan. Promptir: Prompting for all-in-one blind image restoration. *arXiv preprint arXiv:2306.13090*, 2023.

[14] Yitong Jiang, Zhaoyang Zhang, Tianfan Xue, and Jinwei Gu. Autodir: Automatic all-in-one image restoration with latent diffusion. *arXiv preprint arXiv:2310.10123*, 2023.

[15] Ziwei Luo, Fredrik K Gustafsson, Zheng Zhao, Jens Sjölund, and Thomas B Schön. Controlling vision-language models for universal image restoration. *arXiv preprint arXiv:2310.01018*, 2023.

[16] Boyun Li, Xiao Liu, Peng Hu, Zhongqin Wu, Jiancheng Lv, and Xi Peng. All-in-one image restoration for unknown corruption. In *Proceedings of the IEEE/CVF Conference on Computer Vision and Pattern Recognition*, pages 17452–17462, 2022.

[17] Yijun Yang, Angelica I Aviles-Rivero, Huazhu Fu, Ye Liu, Weiming Wang, and Lei Zhu. Video adverse-weather-component suppression network via weather messenger and adversarial backpropagation. In *Proceedings of the IEEE/CVF International Conference on Computer Vision*, pages 13200–13210, 2023.

[18] Yuanshuo Cheng, Mingwen Shao, Yecong Wan, Lixu Zhang, Wangmeng Zuo, and Deyu Meng. Cross-consistent deep unfolding network for adaptive all-in-one video restoration. *arXiv preprint arXiv:2309.01627*, 2023.

[19] Yijun Yang, Hongtao Wu, Angelica I Aviles-Rivero, Yulun Zhang, Jing Qin, and Lei Zhu. Genuine knowledge from practice: Diffusion test-time adaptation for video adverse weather removal. *arXiv preprint arXiv:2403.07684*, 2024.

[20] Muhammad Haris, Gregory Shakhnarovich, and Norimichi Ukita. Recurrent back-projection network for video super-resolution. In *Proceedings of the IEEE/CVF conference on computer vision and pattern recognition*, pages 3897–3906, 2019.

[21] Jose Caballero, Christian Ledig, Andrew Aitken, Alejandro Acosta, Johannes Totz, Zehan Wang, and Wenzhe Shi. Real-time video super-resolution with spatio-temporal networks and motion compensation. In *Proceedings of the IEEE conference on computer vision and pattern recognition*, pages 4778–4787, 2017.

[22] Jianing Deng, Li Wang, Shiliang Pu, and Cheng Zhuo. Spatio-temporal deformable convolution for compressed video quality enhancement. In *Proceedings of the AAAI conference on artificial intelligence*, volume 34, pages 10696–10703, 2020.

[23] Takashi Isobe, Songjiang Li, Xu Jia, Shanxin Yuan, Gregory Slabaugh, Chunjing Xu, Ya-Li Li, Shengjin Wang, and Qi Tian. Video super-resolution with temporal group attention. In *Proceedings of the IEEE/CVF conference on computer vision and pattern recognition*, pages 8008–8017, 2020.

[24] Chao Zhu, Hang Dong, Jinshan Pan, Boyang Liang, Yuhao Huang, Lean Fu, and Fei Wang. Deep recurrent neural network with multi-scale bi-directional propagation for video deblurring. In *Proceedings of the AAAI conference on artificial intelligence*, volume 36, pages 3598–3607, 2022.

[25] Meisong Zheng, Qunliang Xing, Minglang Qiao, Mai Xu, Lai Jiang, Huaida Liu, and Ying Chen. Progressive training of a two-stage framework for video restoration. In *Proceedings of the IEEE/CVF Conference on Computer Vision and Pattern Recognition*, pages 1024–1031, 2022.

[26] Kelvin CK Chan, Xintao Wang, Ke Yu, Chao Dong, and Chen Change Loy. Basicvsr: The search for essential components in video super-resolution and beyond. In *Proceedings of the IEEE/CVF conference on computer vision and pattern recognition*, pages 4947–4956, 2021.

[27] Dequan Wang, Evan Shelhamer, Shaoteng Liu, Bruno Olshausen, and Trevor Darrell. Tent: Fully test-time adaptation by entropy minimization. *arXiv preprint arXiv:2006.10726*, 2020.

[28] Ruoteng Li, Robby T Tan, and Loong-Fah Cheong. All in one bad weather removal using architectural search. In *Proceedings of the IEEE/CVF conference on computer vision and pattern recognition*, pages 3175–3185, 2020.

[29] Wei-Ting Chen, Zhi-Kai Huang, Cheng-Che Tsai, Hao-Hsiang Yang, Jian-Jiun Ding, and Sy-Yen Kuo. Learning multiple adverse weather removal via two-stage knowledge learning and multi-contrastive regularization: Toward a unified model. In *Proceedings of the IEEE/CVF Conference on Computer Vision and Pattern Recognition*, pages 17653–17662, 2022.

[30] Yurui Zhu, Tianyu Wang, Xueyang Fu, Xuanyu Yang, Xin Guo, Jifeng Dai, Yu Qiao, and Xiaowei Hu. Learning weather-general and weather-specific features for image restoration under multiple adverse weather conditions. In *Proceedings of the IEEE/CVF Conference on Computer Vision and Pattern Recognition*, pages 21747–21758, 2023.

[31] Ben Fei, Zhaoyang Lyu, Liang Pan, Junzhe Zhang, Weidong Yang, Tianyue Luo, Bo Zhang, and Bo Dai. Generative diffusion prior for unified image restoration and enhancement. In *Proceedings of the IEEE/CVF Conference on Computer Vision and Pattern Recognition*, pages 9935–9946, 2023.

[32] Jiaqi Ma, Tianheng Cheng, Guoli Wang, Qian Zhang, Xinggang Wang, and Lefei Zhang. Prores: Exploring degradation-aware visual prompt for universal image restoration. *arXiv preprint arXiv:2306.13653*, 2023.

[33] Yuang Ai, Huaibo Huang, Xiaoqiang Zhou, Jiexiang Wang, and Ran He. Multimodal prompt perceiver: Empower adaptiveness generalizability and fidelity for all-in-one image restoration. In *Proceedings of the IEEE/CVF Conference on Computer Vision and Pattern Recognition*, pages 25432–25444, 2024.

[34] Robin Rombach, Andreas Blattmann, Dominik Lorenz, Patrick Esser, and Björn Ommer. High-resolution image synthesis with latent diffusion models. In *Proceedings of the IEEE/CVF conference on computer vision and pattern recognition*, pages 10684–10695, 2022.

[35] Yuanbiao Gou, Haiyu Zhao, Boyun Li, Xinyan Xiao, and Xi Peng. Test-time degradation adaption for open-set image restoration. *arXiv preprint arXiv:2312.02197v3*, 2024.

[36] Wenzhe Shi, Jose Caballero, Ferenc Huszár, Johannes Totz, Andrew P Aitken, Rob Bishop, Daniel Rueckert, and Zehan Wang. Real-time single image and video super-resolution using an efficient sub-pixel convolutional neural network. In *Proceedings of the IEEE conference on computer vision and pattern recognition*, pages 1874–1883, 2016.

[37] Jifeng Dai, Haozhi Qi, Yuwen Xiong, Yi Li, Guodong Zhang, Han Hu, and Yichen Wei. Deformable convolutional networks. In *Proceedings of the IEEE international conference on computer vision*, pages 764–773, 2017.

[38] Alexey Dosovitskiy, Philipp Fischer, Eddy Ilg, Philip Hausser, Caner Hazirbas, Vladimir Golkov, Patrick Van Der Smagt, Daniel Cremers, and Thomas Brox. Flownet: Learning optical flow with convolutional networks. In *Proceedings of the IEEE international conference on computer vision*, pages 2758–2766, 2015.

[39] Jiezhang Cao, Qin Wang, Jingyun Liang, Yulun Zhang, Kai Zhang, Radu Timofte, and Luc Van Gool. Learning task-oriented flows to mutually guide feature alignment in synthesized and real video denoising. *arXiv preprint arXiv:2208.11803*, 2022.

[40] Kai Zhang, Yawei Li, Jingyun Liang, Jiezhang Cao, Yulun Zhang, Hao Tang, Deng-Ping Fan, Radu Timofte, and Luc Van Gool. Practical blind image denoising via swin-conv-unet and data synthesis. *Machine Intelligence Research*, 20(6):822–836, 2023.

[41] F. Perazzi, J. Pont-Tuset, B. McWilliams, L. Van Gool, M. Gross, and A. Sorkine-Hornung. A benchmark dataset and evaluation methodology for video object segmentation. In *Computer Vision and Pattern Recognition*, 2016.

[42] Matias Tassano, Julie Delon, and Thomas Veit. Dvdnet: A fast network for deep video denoising. In *2019 IEEE International Conference on Image Processing (ICIP)*, pages 1805–1809. IEEE, 2019.

[43] Anurag Ranjan and Michael J Black. Optical flow estimation using a spatial pyramid network. In *Proceedings of the IEEE conference on computer vision and pattern recognition*, pages 4161–4170, 2017.

[44] Simon Niklaus. A reimplementation of SPyNet using PyTorch. `https://github.com/sniklaus/pytorch-spynet`, 2018.

[45] Adam Paszke, Sam Gross, Francisco Massa, Adam Lerer, James Bradbury, Gregory Chanan, Trevor Killeen, Zeming Lin, Natalia Gimelshein, Luca Antiga, et al. Pytorch: An imperative style, high-performance deep learning library. *Advances in neural information processing systems*, 32, 2019.

[46] Pierre Charbonnier, Laure Blanc-Feraud, Gilles Aubert, and Michel Barlaud. Two deterministic half-quadratic regularization algorithms for computed imaging. In *Proceedings of 1st international conference on image processing*, volume 2, pages 168–172. IEEE, 1994.

[47] Diederik P Kingma and Jimmy Ba. Adam: A method for stochastic optimization. *arXiv preprint arXiv:1412.6980*, 2014.

[48] Ilya Loshchilov and Frank Hutter. Sgdr: Stochastic gradient descent with warm restarts. *arXiv preprint arXiv:1608.03983*, 2016.

[49] Ozan Özdenizci and Robert Legenstein. Restoring vision in adverse weather conditions with patch-based denoising diffusion models. *IEEE Transactions on Pattern Analysis and Machine Intelligence*, 2023.

[50] Jeya Maria Jose Valanarasu, Rajeev Yasarla, and Vishal M Patel. Transweather: Transformer-based restoration of images degraded by adverse weather conditions. In *Proceedings of the IEEE/CVF Conference on Computer Vision and Pattern Recognition*, pages 2353–2363, 2022.

## A  Appendix

In this section, we first present more details about degradations in the data synthesis approach. Then, we present more qualitative results to demonstrate the effectiveness of our AverNet. Finally, we discuss the broader impacts and limitations of this work.

### A.1 Degradations in Video Synthesis

To address data scarcity issue for studying time-varying unknown degradations (TUD), we propose a new approach based on the degradation model in [40, 39] to synthesize corrupted-clean video pairs that contain TUD. In the synthesis pipeline, the video clips are degraded by three major categories of degradations, *i.e.*, noise, blur, and compression. In the following, we will detail the types and parameters of each degradation within these categories.

**Noise.** In the pipeline, there are three kinds of common noise, *i.e.*, Gaussian noise, Poisson noise, and speckle noise. For Gaussian noise and speckle noise, the noise levels are both uniformly sampled from [10, 15]. The Poisson noise is mathematically modeled as

$$n \sim \mathcal{P}(10^\alpha \times x)/10^\alpha - x, \tag{12}$$

where the $\alpha$ is uniformly sampled from [2,4].

**Blur.** There are two types of blur in pipeline, *i.e.*, Gaussian blur and resizing blur, which usually appear in action videos and Internet videos. For Gaussian blur, the kernel size is uniformly sampled from {3,5,7}, and the kernel is randomly chosen from {'iso', 'aniso', 'generalized_iso', 'generalized_aniso', 'plateau_iso', 'plateau_aniso'} with the probabilities of {0.405, 0.225, 0.108, 0.027, 0.108, 0.027}. For resizing blur, the resize scale and the interpolation mode is uniformly sampled from [0.5, 2] and {'bilinear', 'area', 'bicubic'}, respectively.

**Compression.** This degradation includes JPEG and video compression. For JPEG compression, the quality factor is randomly chosen from {20,30,40}. For video compression, the codecs and bitrate are randomly selected from {'libx264', 'h264', ''mpeg4'} and [1e4, 1e5], respectively.

### A.2 Qualitative Results on Videos with TUD

In addition to the qualitative results presented in the main body of the paper, we show more results on the datasets with TUD. To be specific, Fig. 6 and 7 present the qualitative results on DAVIS-test [41] and Set8 [42] datasets with variation intensity $t = 6$. Moreover, Fig. 8 shows the qualitative results in the noise&blur combination on DAVIS-test. As shown in Fig. 6 and 7, BasicVSR++ [2] and Shift-Net [5] leave noise and artifacts in the frames. Furthermore, RVRT [4] produces frames with significant color distortion. In contrast, our method yields results with finer details and less artifacts. From Fig. 8, one could observe that the results of all-in-one image restoration methods are blurry, and video restoration methods BasicVSR++ and RVRT leave artifacts in the results. In contrast, the results of our method have clearer outlines and are closer to the GT.

### A.3 Broader Impact

In this section, we discuss the impact of our AverNet in a broader vision. Generally, AverNet is the first all-in-one solution to recover videos that contain time-varying unknown degradations, which are prevalent in real-world scenarios. Therefore, it may have multiple applications such as film restoration, surveillance video enhancement, and medical image restoration. However, the videos restored by AverNet may not have the permission of the original copyright holder, thereby infringing the rights of others. Moreover, the training and testing of the model consume a lot of electricity, which causes carbon emissions.

### A.4 Limitation

In addition to the three types of degradations studied in this paper, there are more degradations such as rain, haze and snow that are worth exploring under the TUD setting. Besides, the training data for AverNet is based on the aforementioned video synthesis approach, which generates videos with time-varying unknown degradations closing to real-world scenarios. However, the corruption in the real-world videos are more complex and hard to be simulated. Therefore, in real-world applications, our AverNet needs further validation and improvement.

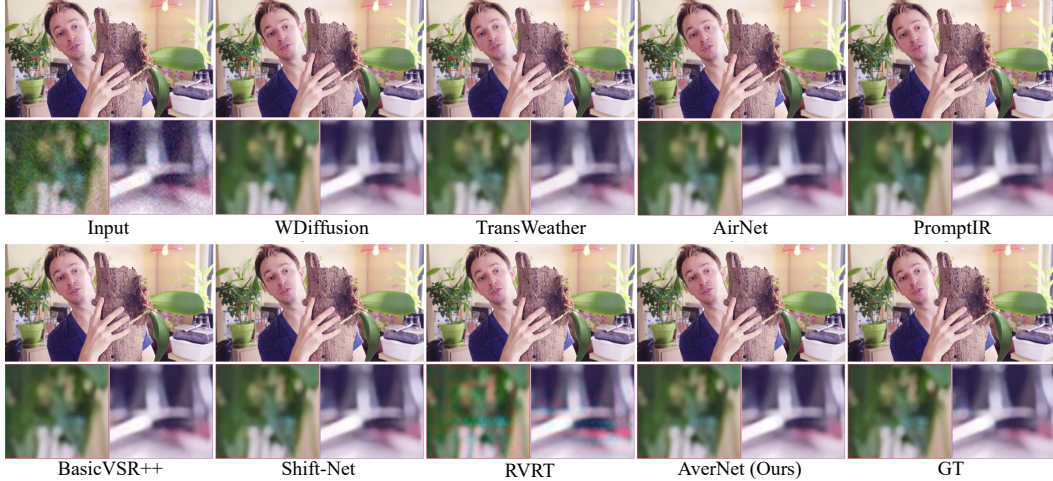

Figure 6: Qualitative results on the "orchid" video from DAVIS ($t = 6$), from which one could see that existing methods leave noise or artifacts in the results. In contrast, the results of our method have less artifacts and finer details.

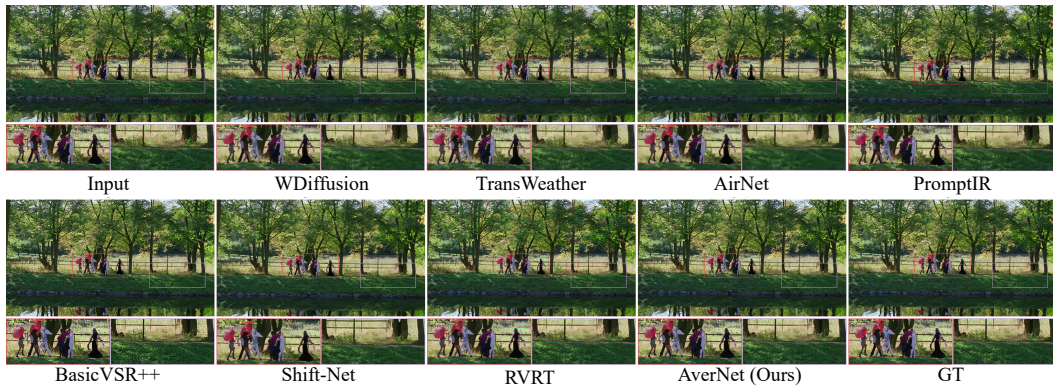

Figure 7: Qualitative results on the "park_joy" video from Set8 ($t = 6$), from which one could observe that existing methods yield blurry or distorted results. In contrast, the results of our methods are clearer and closer to the GT.

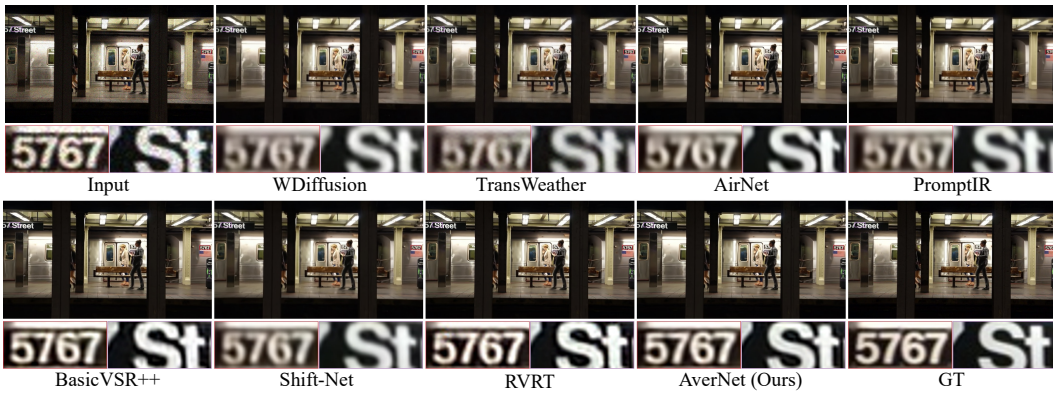

Figure 8: Qualitative results on the "subway" video from DAVIS-test in the noise&blur degradation combination, from which one could observe that the results of existing methods are blurry. In contrast, the results of our method have clearer outlines and tones that are more similar to the GT.

